# Active Data Clustering

**Thomas Hofmann**
Center for Biological and Computational Learning, MIT
Cambridge, MA 02139, USA, hofmann@ai.mit.edu

**Joachim M. Buhmann**
Institut für Informatik III, Universität Bonn
Römerstraße 164, D-53117 Bonn, Germany, jb@cs.uni-bonn.de

## Abstract

*Active data clustering* is a novel technique for clustering of proximity data which utilizes principles from sequential experiment design in order to interleave data generation and data analysis. The proposed active data sampling strategy is based on the *expected value of information*, a concept rooting in statistical decision theory. This is considered to be an important step towards the analysis of large-scale data sets, because it offers a way to overcome the inherent data sparseness of proximity data. We present applications to unsupervised texture segmentation in computer vision and information retrieval in document databases.

## 1 Introduction

*Data clustering* is one of the core methods for numerous tasks in pattern recognition, exploratory data analysis, computer vision, machine learning, data mining, and in many other related fields. Concerning the data representation it is important to distinguish between *vectorial data* and *proximity data*, cf. [Jain, Dubes, 1988]. In vectorial data each measurement corresponds to a certain 'feature' evaluated at an external scale. The elementary measurements of proximity data are, in contrast, (dis-)similarity values obtained by comparing pairs of entities from a given data set. Generating proximity data can be advantageous in cases where 'natural' similarity functions exist, while extracting features and supplying a meaningful vector-space metric may be difficult. We will illustrate the data generation process for two exemplary applications: unsupervised segmentation of textured images and data mining in a document database.

Textured image segmentation deals with the problem of partitioning an image into regions of homogeneous texture. In the unsupervised case, this has to be achieved on

the basis of texture similarities without prior knowledge about the occuring textures. Our approach follows the ideas of [Geman *et al.*, 1990] to apply a *statistical test* to empirical distributions of image features at different sites. Suppose we decided to work with the gray-scale representation directly. At every image location $p = (x, y)$ we consider a local sample of gray-values, e.g., in a squared neighborhood around $p$. Then, the dissimilarity between two sites $p_i$ and $p_j$ is measured by the significance of rejecting the hypothesis that both samples were generated from the same probability distribution. Given a suitable binning $(t_k)_{1 \leq k \leq R}$ and histograms $f_i$, $f_j$, respectively, we propose to apply a $\chi^2$-test, i.e.,

$$D_{ij} = \sum_k \frac{(f_i(t_k) - f_{ij}(t_k))^2}{f_{ij}(t_k)}, \quad \text{with } f_{ij}(t_k) = \frac{f_i(t_k) + f_j(t_k)}{2} \; . \tag{1}$$

In fact, our experiments are based on a multi-scale Gabor filter representation instead of the raw data, cf. [Hofmann *et al.*, 1997] for more details. The main advantage of the similarity-based approach is that it does not reduce the distributional information, e.g., to some simple first and second order statistics, *before* comparing textures. This preserves more information and also avoids the ad hoc specification of a suitable metric like a weighted Euclidean distance on vectors of extracted moment statistics.

As a second application we consider structuring a database of documents for improved information retrieval. Typical *measures of association* are based on the number of shared index terms [Van Rijsbergen, 1979]. For example, a document is represented by a (sparse) binary vector $B$, where each entry corresponds to the occurrence of a certain index term. The dissimilarity can then be defined by the cosine measure

$$D_{ij} = 1 - (B_i^t B_j / \sqrt{|B_i||B_j|}) \; . \tag{2}$$

Notice, that this measure (like many other) may violate the triangle inequality.

## 2   Clustering Sparse Proximity Data

In spite of potential advantages of similarity-based methods, their major drawback seems to be the scaling behavior with the number of data: given a dataset with $N$ entities, the number of potential pairwise comparisons scales with $\mathcal{O}(N^2)$. Clearly, it is prohibitive to exhaustively perform or store all dissimilarities for large datasets, and the crucial problem is how to deal with this unavoidable data sparseness. More fundamentally, it is already the data generation process which has to solve the problem of *experimental design*, by selecting a subset of pairs $(i, j)$ for evaluation. Obviously, a meaningful selection strategy could greatly profit from any knowledge about the grouping structure of the data. This observation leads to the concept of performing a *sequential* experimental design which interleaves the data clustering with the data acquisition process. We call this technique *active data clustering*, because it actively selects new data, and uses tentative knowledge to estimate the relevance of missing data. It amounts to inferring from the available data not only a grouping structure, but also to learn which future data is most relevant for the clustering problem. This fundamental concept may also be applied to other unsupervised learning problems suffering from data sparseness.

The first step in deriving a clustering algorithm is the specification of a suitable objective function. In the case of similarity-based clustering this is not at all a trivial problem and we have systematically developed an axiomatic approach based on invariance and robustness principles [Hofmann *et al.*, 1997]. Here, we can only

give some informal justifications for our choice. Let us introduce indicator functions to represent data partitionings, $M_{i\nu}$ being the indicator function for entity $o_i$ belonging to cluster $C_\nu$. For a given number $K$ of clusters, all Boolean functions are summarized in terms of an assignment matrix $\mathbf{M} \in \{0,1\}^{N \times K}$. Each row of $\mathbf{M}$ is required to sum to one in order to guarantee a unique cluster membership. To distinguish between known and unknown dissimilarities, index sets or *neighborhoods* $\mathcal{N} = (\mathcal{N}_1, \dots, \mathcal{N}_N)$ are introduced. If $j \in \mathcal{N}_i$ this means the value of $D_{ij}$ is available, otherwise it is not known. For simplicity we assume the dissimilarity measure (and in turn the neighborhood relation) to be symmetric, although this is not a necessary requirement. With the help of these definition the proposed criterion to assess the quality of a clustering configuration is given by

$$\mathcal{H}(\mathbf{M}; \mathbf{D}, \mathcal{N}) = \sum_{i=1}^{N} \sum_{\nu=1}^{K} M_{i\nu} d_{i\nu}, \quad d_{i\nu} = \frac{\sum_{j \in \mathcal{N}_i} M_{j\nu} D_{ij}}{\sum_{j \in \mathcal{N}_i} M_{j\nu}} . \tag{3}$$

$\mathcal{H}$ additively combines contributions $d_{i\nu}$ for each entity, where $d_{i\nu}$ corresponds to the average dissimilarity to entities belonging to cluster $C_\nu$. In the sparse data case, averages are restricted to the fraction of entities with known dissimilarities, i.e., the subset of entities belonging to $C_\nu \cap \mathcal{N}_i$.

## 3   Expected Value of Information

To motivate our active data selection criterion, consider the simplified sequential problem of inserting a new entity (or object) $o_N$ to a database of $N - 1$ entities with a given fixed clustering structure. Thus we consider the decision problem of optimally assigning the new object to one of the $K$ clusters. If all dissimilarities between objects $o_i$ and object $o_N$ are known, the optimal assignment only depends on the average dissimilarities to objects in the different clusters, and hence is given by

$$M_{N\alpha^*} = 1 \iff \alpha^* = \arg\min_\nu d^*_{N\nu}, \quad \text{where} \quad d^*_{N\nu} = \frac{\sum_{j=1}^{N-1} M_{j\nu} D_{Nj}}{\sum_{j=1}^{N-1} M_{j\nu}} . \tag{4}$$

For incomplete data, the total population averages $d^*_{N\nu}$ are replaced by point estimators $d_{N\nu}$ obtained by restricting the sums in (4) to $\mathcal{N}_N$, the neighborhood of $o_N$. Let us furthermore assume we want to compute a fixed number $L$ of dissimilarities before making the terminal decision. If the entities in each cluster are not further distinguished, we can pick a member at random, once we have decided to sample from a cluster $C_\nu$. The selection problem hence becomes equivalent to the problem of optimally distributing $L$ measurements among $K$ populations, such that the risk of making the wrong decision based on the resulting estimates $d_{N\nu}$ is minimal. More formally, this risk is given by $\mathcal{R} = d^*_{N\alpha} - d^*_{N\alpha^*}$, where $\alpha$ is the decision based on the subpopulation estimates $\{d_{N\nu}\}$ and $\alpha^*$ is the true optimum.

To model the problem of selecting an optimal experiment we follow the Bayesian approach developed by Raiffa & Schlaifer [Raiffa, Schlaifer, 1961] and compute the so-called *Expected Value of Sampling Information* (EVSI). As a fundamental step this involves the calculation of distributions for the quantities $d_{N\nu}$. For reasons of computational efficiency we are assuming that dissimilarities resulting from a comparison with an object in cluster $C_\nu$ are normally distributed[1] with mean $d^*_{N\nu}$ and variance $\sigma^{*2}_{N\nu}$. Since the variances are nuisance parameters the risk function $\mathcal{R}$ does not depend on, it suffices to calculate the marginal distribution of

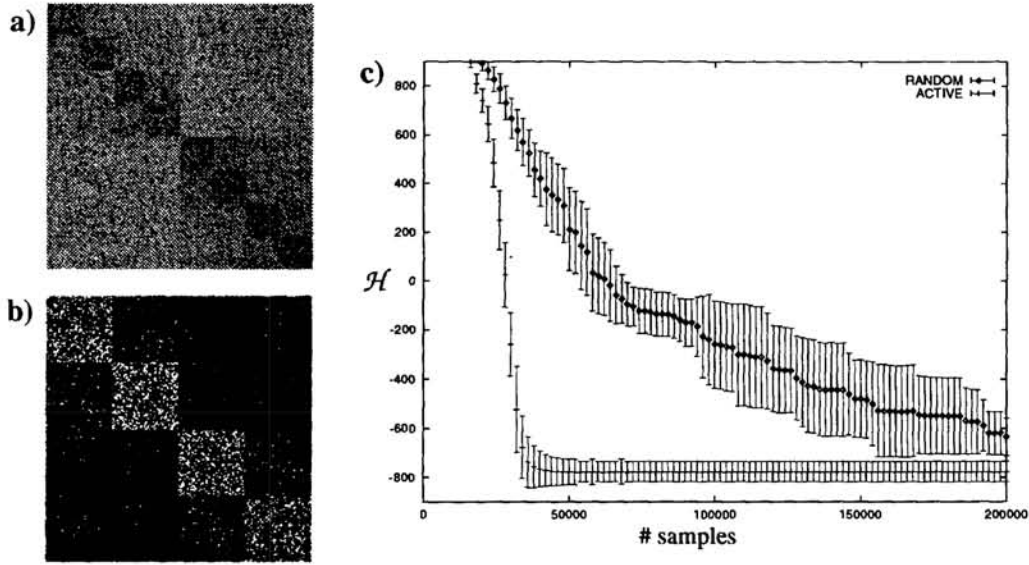

Figure 1: (a) Gray-scale visualization of the generated proximity matrix ($N = 800$). Dark/light gray values correspond to low/high dissimilarities respectively, $D_{ij}$ being encoded by pixel $(i, j)$. (b) Sampling snapshot for active data clustering after 60000 samples, queried values are depicted in white. (c) Costs evaluated on the **complete** data for sequential active and random sampling.

$d_{N\nu}^*$. For the class of statistical models we will consider in the sequel the empirical mean $d_{N\nu}$, the unbiased variance estimator $\sigma_{N\nu}^2$ and the sample size $m_{N\nu}$ are a sufficient statistic. Depending on these empirical quantities the marginal posterior distribution of $d_{N\nu}^*$ for uninformative priors is a Student $t$ distribution with $t = \sqrt{m_{N\nu}}(d_{N\nu}^* - d_{N\nu})/\sigma_{N\nu}$ and $m_{N\nu} - 1$ degrees of freedom. The corresponding density will be denoted by $f_\nu(d_{N\nu}^* | d_{N\nu}, \sigma_{N\nu}^2, m_{N\nu})$. With the help of the posterior densities $f_\nu$ we define the *Expected Value of Perfect Information* (EVPI) after having observed $(d_{N\nu}, \sigma_{N\nu}^2, m_{N\nu})$ by

$$\text{EVPI} = \int_{-\infty}^{+\infty} \cdots \int_{-\infty}^{+\infty} \max_\nu \{d_{N\alpha}^* - d_{N\nu}^*\} \prod_{\nu=1}^{K} f_\nu(d_{N\nu}^* | d_{N\nu}, \sigma_{N\nu}^2, m_{N\nu}) \, d\, d_{N\nu}^*, \qquad (5)$$

where $\alpha = \arg\min_\nu d_{N\nu}$. The EVPI is the loss one expects to incur by making the decision $\alpha$ based on the incomplete information $\{d_{N\nu}\}$ instead of the optimal decision $\alpha^*$, or, put the other way round, the expected gain we would obtain if $\alpha^*$ was revealed to us.

In the case of experimental design, the main quantity of interest is not the EVPI but the *Expected Value of Sampling Information* (EVSI). The EVSI quantifies how much gain we are expecting from additional data. The outcome of additional experiments can only be anticipated by making use of the information which is already available. This is known as *preposterior analysis*. The linearity of the utility measure implies that it suffices to calculate averages with respect to the *preposterous* distribution [Raiffa, Schlaifer, 1961, Chapter 5.3]. Drawing $m_{N\nu}^+$ additional samples from the $\nu$-th population, and averaging possible outcomes with the (prior) distribution $f_\nu(d_{N\nu}^* | d_{N\nu}, \sigma_{N\nu}^2, m_{N\nu})$ will not affect the unbiased estimates $d_{N\nu}, \sigma_{N\nu}^2$, but only increase the number of samples $m_{N\nu} \to m_{N\nu} + m_{N\nu}^+$. Thus, we can compute the EVSI from (5) by replacing the prior densities with its preposterous counterparts.

To evaluate the $K$-dimensional integral in (5) or its EVSI variant we apply Monte-Carlo techniques, sampling from the Student $t$ densities using Kinderman's re-

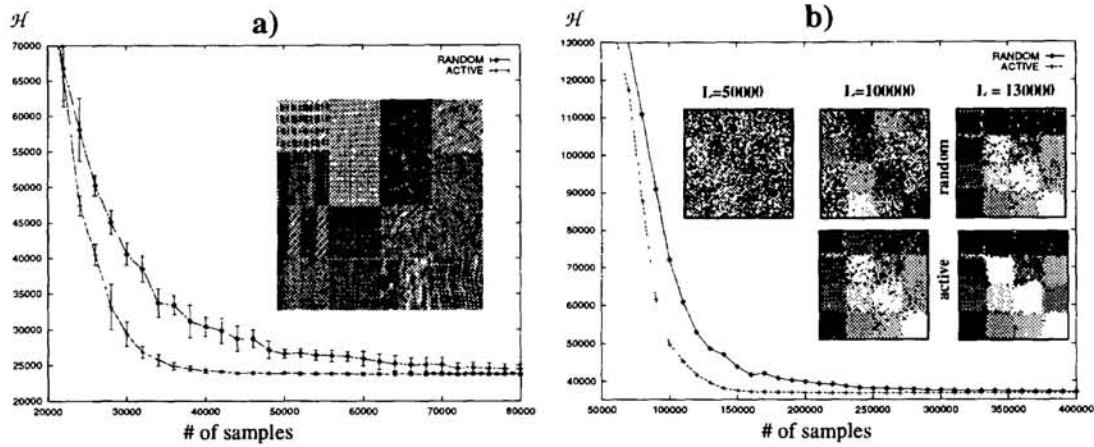

Figure 2: (a) Solution quality for active and random sampling on data generated from a mixture image of 16 Brodatz textures ($N = 1024$). (b) Cost trajectories and segmentation results for an active and random sampling example run ($N = 4096$).

jection sampling scheme, to get an empirical estimate of the random variable $\psi_\alpha(d^*_{N1}, \ldots, d^*_{NK}) = \max_\nu \{d^*_{N\alpha} - d^*_{N\nu}\}$. Though this enables us in principle to approximate the EVSI of any possible experiment, we cannot efficiently compute it for all possible ways of distributing the $L$ samples among $K$ populations. In the large sample limit, however, the EVSI becomes a concave function of the sampling sizes. This motivates a greedy design procedure of drawing new samples incrementally one by one.

## 4   Active Data Clustering

So far we have assumed the assignments of all but one entity $\mathbf{o}_N$ to be given in advance. This might be realistic in certain on-line applications, but more often we want to simultaneously find assignments for all entities in a dataset. The active data selection procedure hence has to be combined with a recalculation of clustering solutions, because additional data may help us not only to improve our terminal decision, but also with respect to our sampling strategy. A local optimization of $\mathcal{H}$ for assignments of a single object $\mathbf{o}_i$ can rely on the quantities

$$g_{i\nu} = \sum_{j \in \mathcal{N}_i} \left[ \frac{1}{n_{i\nu}} + \frac{1}{n_{j\nu}^{+i}} \right] M_{j\nu} D_{ij} - \sum_{j \in \mathcal{N}_i} \frac{1}{n_{j\nu}^{+i} n_{j\nu}^{-i}} \sum_{k \in \mathcal{N}_j - \{i\}} M_{j\nu} M_{k\nu} D_{jk} , \quad (6)$$

where $n_{j\nu} = \sum_{j \in \mathcal{N}_i} M_{j\nu}$, $n_{j\nu}^{-i} = n_{j\nu} - M_{i\nu}$, and $n_{j\nu}^{+i} = n_{j\nu}^{-i} + 1$, by setting $M_{i\alpha} = 1 \iff \alpha = \arg\min_\nu g_{i\nu} = \arg\min_\nu \mathcal{H}(\mathbf{M}|M_{i\nu} = 1)$, a claim which can be proved by straightforward algebraic manipulations (cf. [Hofmann et al., 1997]). This effectively amounts to a cluster readjustment by reclassification of objects. For additional evidence arising from new dissimilarities, one thus performs local reassignments, e.g., by cycling through all objects in random order, until no assignment is changing.

To avoid unfavorable local minima one may also introduce a computational temperature $T$ and utilize $\{g_{i\nu}\}$ for simulated annealing based on the Gibbs sampler [Geman, Geman, 1984], $P\{M_{i\alpha} = 1\} = \exp\left[-\frac{1}{T} g_{i\alpha}\right] / \sum_{\nu=1}^{K} \exp\left[-\frac{1}{T} g_{i\nu}\right]$. Alternatively, Eq. (6) may also serve as the starting point to derive mean-field equations in a deterministic annealing framework, cf. [Hofmann, Buhmann, 1997]. These local

| 1 | 2 | 3 | 4 | 5 | 6 | 7 | 8 | 9 | 10 |
|---|---|---|---|---|---|---|---|---|---|
| cluster | cluster | cluster | cluster | task | cluster | cluster | model | fuzzi | network |
| model | state | atom | algorithm | schedul | structur | object | cluster | cluster | cluster |
| distribu | sup | result | propos | cluster | method | approach | method | algorithm | neural |
| process | particl | temperatur | method | algorithm | base | algorithm | object | data | learn |
| studi | studi | degre | new | graph | gener | base | data | method | algorithm |
| cloud | sup | alloi | speech | schedul | loop | user | model | fuzzi | neural |
| fractal | alpha | atom | continu | task | video | queri | context | membership | network |
| event | state | ion | error | placem | famili | access | decision | rule | competit |
| random | particl | electron | construct | connect | softwar | softwar | manufactur | control | selforgan |
| particl | interac | temperatur | speaker | qualiti | variabl | placem | physical | identif | learn |

| 11 | 12 | 13 | 14 | 15 | 16 | 17 | 18 | 19 | 20 |
|---|---|---|---|---|---|---|---|---|---|
| algorithm | algorithm | cluster | method | cluster | robust | imag | cluster | model | galaxi |
| problem | cluster | data | docum | cluster | cluster | cluster | data | cluster | cluster |
| cluster | fuzzi | propos | signatur | techniqu | system | segment | algorithm | scale | function |
| method | propos | result | cluster | result | complex | algorithm | set | nonlinear | correl |
| optim | data | method | file | paper | eigenvalu | method | method | simul | redshift |
| heurist | converg | link | docum | visual | uncertainti | pixel | dissimilar | nbodi | hsup |
| solv | cmean | singl | retriev | video | robust | segment | point | gravit | redshift |
| tool | algorithm | method | previou | target | perturb | imag | data | dark | mpc |
| program | fcm | retriev | analyt | processor | bound | motion | center | mass | galaxi |
| machin | criteria | hierarchi | literatur | queri | matrix | color | kmean | matter | survei |

Figure 3: Clustering solution with 20 clusters for 1584 documents on 'clustering'. Clusters are characterized by their 5 most topical and 5 most typical index terms.

optimization algorithms are well-suited for an incremental update after new data has been sampled, as they do not require a complete recalculation from scratch. The probabilistic reformulation in an annealing framework has the further advantage to provide assignment probabilities which can be utilized to improve the randomized 'partner' selection procedure. For any of these algorithms we sequentially update data assignments until a convergence criterion is fulfilled.

## 5 Results

To illustrate the behavior of the active data selection criterion we have run a series of repeated experiments on artificial data. For $N = 800$ the data has been divided into 8 groups of 100 entities. Intra-group dissimilarities have been set to zero, while inter-group dissimilarities were defined hierarchically. All values have been corrupted by Gaussian noise. The proximity matrix, the sampling performance, and a sampling snapshot are depicted in Fig. 1. The sampling exactly performs as expected: after a short initial phase the active clustering algorithm spends more samples to disambiguate clusters which possess a higher mean similarity, while less dissimilarities are queried for pairs of entities belonging to well separated clusters. For this type of structured data the gain of active sampling increases with the depth of the hierarchy. The final solution variance is due to local minima. Remarkably the active sampling strategy not only shows a faster improvement, it also finds on average significantly better solution. Notice that the sampling has been decomposed into stages, refining clustering solutions after sampling of 1000 additional dissimilarities.

The results of an experiment for unsupervised texture segmentation is shown Fig. 2. To obtain a close to optimal solution the active sampling strategy roughly needs less than 50% of the sample size required by random sampling for both, a resolution of $N = 1024$ and $N = 4096$. At a $64 \times 64$ resolution, for $L = 100K, 150K, 200K$ actively selected samples the random strategy needs on average $\bar{L} = 120K, 300K, 440K$ samples, respectively, to obtain a comparable solution quality. Obviously, active sampling can only be successful in an intermediate regime: if too little is known, we cannot infer additional information to improve our sampling, if the sample is large enough to reliably detect clusters, there is no need to sample any more. Yet, this intermediate regime significantly increases with $K$ (and $N$).

Finally, we have clustered 1584 documents containing abstracts of papers with *clustering* as a title word. For $K = 20$ clusters[2] active clustering needed 120000 samples ($< 10\%$ of the data) to achieve a solution quality within 1% of the asymptotic solution. A random strategy on average required 230000 samples. Fig. 3 shows the achieved clustering solution, summarizing clusters by topical (most frequent) and typical (most characteristic) index terms. The found solution gives a good overview over areas dealing with clusters and clustering[3].

## 6   Conclusion

As we have demonstrated, the concept of *expected value of information* fits nicely into an optimization approach to clustering of proximity data, and establishes a sound foundation of active data clustering in statistical decision theory. On the medium size data sets used for validation, active clustering achieved a consistently better performance as compared to random selection. This makes it a promising technique for automated structure detection and data mining applications in large data bases. Further work has to address stopping rules and speed-up techniques to accelerate the evaluation of the selection criterion, as well as a unification with annealing methods and hierarchical clustering.

### Acknowledgments

This work was supported by the Federal Ministry of Education and Science BMBF under grant # 01 M 3021 A/4 and by a M.I.T. Faculty Sponser's Discretionary Fund.

## Footnotes

[1] Other computationally more expensive choices to model within cluster dissimilarities are skewed distributions like the Gamma–distribution.

[2]The number of clusters was determined by a criterion based on complexity costs.

[3]Is it by chance, that 'fuzzy' techniques are 'softly' distributed over two clusters?

## References

[Geman *et al.*, 1990] Geman, D., Geman, S., Graffigne, C., Dong, P. (1990). Boundary Detection by Constrained Optimization. *IEEE Transactions on Pattern Analysis and Machine Intelligence*, **12**(7), 609–628.

[Geman, Geman, 1984] Geman, S., Geman, D. (1984). Stochastic Relaxation, Gibbs Distributions, and the Bayesian Restoration of Images. *IEEE Transactions on Pattern Analysis and Machine Intelligence*, 6(6), 721–741.

[Hofmann, Buhmann, 1997] Hofmann, Th., Buhmann, J. M. (1997). Pairwise Data Clustering by Deterministic Annealing. *IEEE Transactions on Pattern Analysis and Machine Intelligence*, **19**(1), 1–14.

[Hofmann *et al.*, 1997] Hofmann, Th., Puzicha, J., Buhmann, J.M. 1997. Deterministic Annealing for Unsupervised Texture Segmentation. *Pages 213–228 of: Proceedings of the International Workshop on Energy Minimization Methods in Computer Vision and Pattern Recognition.* Lecture Notes in Computer Science, vol. 1223.

[Jain, Dubes, 1988] Jain, A. K., Dubes, R. C. (1988). *Algorithms for Clustering Data.* Englewood Cliffs, NJ 07632: Prentice Hall.

[Raiffa, Schlaifer, 1961] Raiffa, H., Schlaifer, R. (1961). *Applied Statistical Decision Theory.* Cambridge MA: MIT Press.

[Van Rijsbergen, 1979] Van Rijsbergen, C. J. (1979). *Information Retrieval.* Butterworths, London Boston.

